# Concentration Inequalities for the Missing Mass and for Histogram Rule Error

**David McAllester**
Toyota Technological Institute at Chicago
mcallester@tti-c.org

**Luis Ortiz**
University of Pennsylvania
leo@cis.upenn.edu

## Abstract

This paper gives distribution-free concentration inequalities for the missing mass and the error rate of histogram rules. Negative association methods can be used to reduce these concentration problems to concentration questions about independent sums. Although the sums are independent, they are highly heterogeneous. Such highly heterogeneous independent sums cannot be analyzed using standard concentration inequalities such as Hoeffding's inequality, the Angluin-Valiant bound, Bernstein's inequality, Bennett's inequality, or McDiarmid's theorem.

## 1 Introduction

The Good-Turing missing mass estimator was developed in the 1940s to estimate the probability that the next item drawn from a fixed distribution will be an item not seen before. Since the publication of the Good-Turing missing mass estimator in 1953 [9], this estimator has been used extensively in language modeling applications [4, 6, 12]. Recently a large deviation accuracy guarantee was proved for the missing mass estimator [15, 14]. The main technical result is that the missing mass itself concentrates — [15] proves that the probability that missing mass deviates from its expectation by more than $\epsilon$ is at most $e^{-m\epsilon^2/3}$ independent of the underlying distribution. Here we give a simpler proof of the stronger result that the deviation probability is bounded by $e^{-m\epsilon^2}$.

A histogram rule is defined by two things — a given clustering of objects into classes and a given training sample. In a classification setting the histogram rule defined by a given clustering and sample assigns to each cluster the label that occurred most frequently for that cluster in the sample. In a decision-theoretic setting, such as that studied by Ortiz and Kaebling [16], the rule associates each cluster with the action choice of highest performance on the training data for that cluster. We show that the performance of a histogram rule (for a fixed clustering) concentrates near its expectation — the probability that the performance deviates from its expectation by more than $\epsilon$ is bounded by $e^{-m\epsilon^2/9}$ independent of the clustering or the underlying data distribution.

## 2 The Exponential Moment Method

All of the results in this paper are based on the exponential moment method of proving concentration inequalities. The exponential moment was perhaps first used by Bernstein

but was popularized by Chernoff. Let $X$ be any real-valued random variable with finite mean. Let $DP(X, x)$ be $P(X \geq x)$ if $x \geq \mathrm{E}[X]$ and $P(X \leq x)$ is $x < \mathrm{E}[X]$. The following lemma is the central topic of Chernoff's classic paper [5].

**Lemma 1 (Chernoff)** *For any real-valued variable $X$ with finite mean $\mathrm{E}[X]$ we have the following for any $x$ where the "entropy" $S(X, x)$ is defined as below.*

$$DP(X, \ x) \quad \leq \quad e^{-S(X, x)} \tag{1}$$

$$S(X, x) \quad = \quad \sup_{\beta} \ x\beta - \ln Z(X, \ \beta) \tag{2}$$

$$Z(X, \ \beta) \quad = \quad \mathrm{E}\left[e^{\beta X}\right] \tag{3}$$

Lemma 1 follows, essentially, from the observation that for $\beta \geq 0$ we have the following.

$$P(X \geq x) \leq \mathrm{E}\left[e^{\beta(X - x)}\right] = e^{-\beta x}\mathrm{E}\left[e^{\beta X}\right] = e^{-(x\beta - \ln Z(X, \ \beta))} \tag{4}$$

Lemma 1 is called the exponential moment method because of the first inequality in (4). The following two observations provide a simple general tool.

**Observation 2** *Let $s$ be any positive constant satisfying $\ln Z(X, \ \beta) \leq \mathrm{E}[X]\beta + s\beta^2$ for all $\beta \geq 0$. Formula (2) implies that for $\epsilon \geq 0$ we have $S(\mathrm{E}[X] + \epsilon) \geq \epsilon^2/(4s)$.*

**Observation 3** *If $X_1, \ldots, X_n$ are independent then $\ln Z(\sum_i X_i, \ \beta) = \sum_i \ln Z(X_i, \ \beta)$.*

Some further observations also prove useful. Let $X$ be an arbitrary real-valued random variable. For a discrete distribution the Gibbs distribution $P_\beta$ can be defined as follows.

$$P_\beta(X = x) = \frac{1}{Z(X, \ \beta)}P(X = x)e^{\beta x}$$

There exists a unique largest open interval $(\beta_{\min}, \ \beta_{\max})$ (possibly with infinite endpoints) such that for $\beta \in (\beta_{\min}, \beta_{\max})$ we have that $Z(X, \beta)$ is finite. For $\beta \in (\beta_{\min}, \ \beta_{\max})$ we define the expectation of $f(X)$ at inverse temperature $\beta$ as follows.

$$\mathrm{E}_\beta[f(X)] = \frac{1}{Z(X, \ \beta)} \ \mathrm{E}\left[f(X)e^{\beta X}\right] \tag{5}$$

Equation (5) can be taken as the definition of $P_\beta$ for continuous distributions on $X$. For $\beta \in (\beta_{\min}, \ \beta_{\max})$ let $\sigma^2(X, \ \beta)$ be $\mathrm{E}_\beta\left[(X - \mathrm{E}_\beta[X])^2\right]$. The quantity $\sigma^2(X, \ \beta)$ is the Gibbs-variance at inverse temperature $\beta$. For $\beta \in (\beta_{\min}, \beta_{\max})$ we let $KL(P_\beta \| P)$ denote the KL-divergence from $P_\beta$ to $P$ which can be written as follows.

$$KL(P_\beta \| P) = \mathrm{E}_\beta[X]\beta - \ln Z(X, \ \beta) \tag{6}$$

Let $(x_{\min}, \ x_{\max})$ be the smallest open interval containing all values of the form $\mathrm{E}_\beta[X]$ for $\beta \in (\beta_{\min}, \ \beta_{\max})$. If the open interval $(x_{\min}, \ x_{\max})$ is not empty then $\mathrm{E}_\beta[X]$ is a monotonically increasing function of $\beta \in (\beta_{\min}, \ \beta_{\max})$. For $x \in (x_{\min}, \ x_{\max})$ define $\beta(x)$ to be the unique value $\beta$ satisfying $\mathrm{E}_\beta[X] = x$. For any continuous function $f$ we now define the double integral $\int\!\!\int_a^x f(s) \ d^2s$ to be the function $F(x)$ satisfying $F(a) = 0$, $F'(a) = 0$, and $F''(x) = f(x)$ where $F'(x)$ and $F''(x)$ are the first and second derivatives of $F$ respectively. We now have the following general theorem.

**Theorem 4** *For any real-valued variable $X$, any $x \in (x_{\min}, x_{\max})$, and $\beta \in (\beta_{\min}, \ \beta_{\max})$ we have the following.*

$$S(X, \ x) \quad = \quad x\beta(x) - \ln Z(X, \ \beta(x)) \tag{7}$$

$$= KL(P_{\beta(x)}||P) \tag{8}$$

$$= \iint_{\mathrm{E}[X]}^{x} \frac{d^2 z}{\sigma^2(X,\ \beta(z))} \tag{9}$$

$$\ln Z(X,\ \beta) = \mathrm{E}_0[X]\beta + \iint_0^\beta \sigma^2(X,\ \gamma)d^2\gamma \tag{10}$$

Formula (9) can be clarified by noting that for $|x - \mathrm{E}[X]|$ small we have the following.

$$S(X,x) = \iint_{\mathrm{E}[X]}^{x} \frac{d^2 z}{\sigma^2(X,\ \beta(z))} \approx \frac{(x - \mathrm{E}[X])^2}{2\sigma^2(X,\ 0)}$$

Formula (7) is proved by showing that $\beta(x)$ is the the optimal $\beta$ in (2). Up to sign conventions (7) is the equation for physical entropy in statistical mechanics. Equation (8) follows from (7) and (6). Equations (9) and (10) then follow from well known equations of statistical mechanics. An implicit derivation of (9) and (10) can be found in section six of Chernoff's original paper [5].

As a simple example of the use of (9), we derive Hoeffding's inequality. Consider a sum $X = \sum_{i=1}^{n} X_i$ where the $X_i$ are independent and $X_i$ is bounded to an interval of width $b_i$. Note that each $X_i$ remains bounded to this interval at all values of $\beta$. Hence $\sigma^2(X_i,\ \beta) \leq \frac{b_i^2}{4}$. We then have that $\sigma^2(X,\ \beta) \leq \frac{1}{4}\sum_{i=1}^{n} b_i^2$. Hoeffding's inequality now follows from (1) and (9).

## 3   Negative Association

The analysis of the missing mass and histogram rule error involve sums of variables that are not independent. However, these variables are negatively associated — an increase in one variable is associated with decreases in the other variables. Formally, a set of real-valued random variables $X_1, \ldots, X_n$ is negatively associated if for any two disjoint subsets $I$ and $J$ of the integers $\{1,\ \ldots,\ n\}$, and any two non-decreasing, or any two non-increasing, functions $f$ from $R^{|I|}$ to $R$ and $g$ from $R^{|J|}$ to $R$ we have the following.

$$\mathrm{E}[f(X_i,\ i \in I)g(X_j,\ j \in J)] \leq \mathrm{E}[f(X_i,\ i \in I)]\,\mathrm{E}[g(X_j,\ j \in J)]$$

Dubhasi and Ranjan [8] give a survey of methods for establishing and using negative association. This section states some basic facts about negative association.

**Lemma 5** *Let $X_1, \ldots, X_n$ be any set of negatively associated variables. Let $X_1', \ldots, X_n'$ be independent shadow variables, i.e., independent variables such that $X_i'$ is distributed identically to $X_i$. Let $X = \sum_i X_i$ and $X' = \sum_i X_i'$. For any set of negatively associated variables we have $S(X,x) \geq S(X',x)$.*

**Lemma 6** *Let $S$ be any sample of $m$ items (ball throws) drawn IID from a fixed distribution on the integers (bins) $\{1,\ \ldots,\ V\}$. Let $c(i)$ be the number of times integer $i$ occurs in the sample. The variables $c(1),\ \ldots,\ c(V)$ are negatively associated.*

**Lemma 7** *For any negatively associated variables $X_1, \ldots, X_n$, and any non-decreasing functions $f_1, \ldots, f_n$, we have that the quantities $f_1(X_1), \ldots, f_n(X_n)$ are negatively associated. This also holds if the functions $f_i$ are non-increasing.*

**Lemma 8** *Let* $X_1$, ..., $X_n$ *be a negatively associated set of variables. Let* $Y_1$ ..., $Y_n$ *be 0-1 (Bernoulli) variables such that* $Y_i$ *is a stochastic function of* $X_i$, *i.e.,* $P(Y_i = 1 \mid X_1, \ldots, X_n) = P(Y_i = 1 \mid X_i)$. *If* $P(Y_1 = 1 \mid X_i)$ *is a non-decreasing function of* $X_i$ *then* $Y_1$, ..., $Y_n$ *are negatively associated. This also holds if* $P(Y_i = 1 \mid X_i)$ *is non-increasing.*

## 4 The Missing Mass

Suppose that we draw words (or any objects) independently from a fixed distribution over a countable (but possibly infinite) set of words. We let the probability of drawing word $w$ be denoted as $P_w$. For a sample $S$ of $m$ draws the missing mass of $S$, denoted $X$, is the total probability mass of the items not occurring in the sample, i.e. $X = \sum_{w \notin S} P_w$.

**Theorem 9** *For the missing mass* $X$ *as defined above, and for* $\epsilon \geq 0$, *we have the following.*

$$S(X, \ \mathrm{E}[X] - \epsilon) \ \geq \ \frac{4}{3} m \epsilon^2 \tag{11}$$

$$S(X, \ \mathrm{E}[X] + \epsilon) \ \geq \ m \epsilon^2 \tag{12}$$

To prove theorem 9 let $X_w$ be a Bernoulli variable which is 1 if word $w$ does *not* occur in the sample and 0 otherwise. The missing mass can now be written as $X = \sum_w P_w X_w$. The variables $X_w$ are monotonic functions of the word counts so by lemmas 6 and 7 we have that the $X_w$ are negatively associated. By lemma 5 we can then assume that the variables $X_w$ are independent. The analysis of this independent sum uses the following general concentration inequalities for independent sums of bounded variables.

**Lemma 10** *Let* $X = \sum_{i=1}^{V} b_i X_i$ *where* $X_1$, ..., $X_V$ *are independent random variables with* $X_i \in [0, 1]$ *and each* $b_i$ *is a non-negative constant. Let* $Q_i$ *be* $\mathrm{E}[X_i]$. *For* $\epsilon \geq 0$ *we have the following.*

$$S(X, \mathrm{E}[X] - \epsilon) \ \geq \ \frac{\epsilon^2}{2 \sum_{i=1}^{V} Q_i b_i^2} \tag{13}$$

$$S(X, \mathrm{E}[X] + \epsilon) \ \geq \ \frac{\epsilon^2}{\sum_{i=1}^{V} \frac{b_i^2}{\ln \frac{1}{Q_i}}} \tag{14}$$

Before proving (13) and (14) we first show how (13) and (14) imply (11) and (12) respectively. For the missing mass $X = \sum_w P_w X_w$ we have the following.

$$Q_w = P(X_w = 1) = (1 - P_w)^m \leq e^{-P_w m}$$

To prove (11) we note that formula (13) implies the following where we use the fact that for $x \geq 0$ we have $e^{-x} \leq 1/(ex)$.

$$S(X, \mathrm{E}[X] - \epsilon) \geq \frac{\epsilon^2}{2 \sum_w P_w^2 e^{-P_w m}} \ \geq \ \frac{\epsilon^2}{2 \sum_w P_w/(em)} \ = \ \frac{e}{2} m \epsilon^2$$

To prove (12) we note that formula (14) implies the following.

$$S(X, \mathrm{E}[X] + \epsilon) \ \geq \ \frac{\epsilon^2}{\sum_w P_w^2 / \ln \frac{1}{Q_w}} \ \geq \ \frac{\epsilon^2}{\sum_w P_w/m} \ = \ m \epsilon^2$$

We now compare (13) and (14) to other well known bounds. Hoeffding's inequality [11] yields the following.

$$S(X, \mathrm{E}[X] + \epsilon) \geq \frac{2\epsilon^2}{\sum_{i=1}^{V} b_i^2} \qquad (15)$$

In the missing mass application we have that $\sum_{i=1}^{V} b_i^2$ can be $\Omega(1)$ which fails to yield (12). The Srivistav-Stangier bound [17], which itself an improvement on the Angluin-Valiant bound [1, 10], yields the following for $0 \leq \epsilon \leq \overline{X}$ where $b_{\max}$ is $\max_i b_i$.

$$S(X, \mathrm{E}[X] + \epsilon) \geq \frac{\epsilon^2}{3b_{\max} \sum_{i=1}^{V} b_i Q_i} \qquad (16)$$

It is possible to show that in the missing mass application $b_{\max} \sum_{i=1}^{V} b_i Q_i$ can be $\Omega(1)$ so this bound does not handle the missing mass. A weaker version of the lower-deviation inequality (13) can be derived from Bernstein's inequality [3] (see [7]). However, neither Bernstein''s inequality nor Bennett's inequality [2] can handle the upward deviation of the missing mass.

To prove (13) and (14) we first note the following lemma.

**Lemma 11** *Let $X$ be a random variable with $X \in [0, 1]$ and let $X' \in \{0, 1\}$ be a Bernoulli variable with $\mathrm{E}[X'] = \mathrm{E}[X]$. For any such variables $X$ and $X'$ and any $\beta$ and constant $a$ we have the following.*
$$\ln Z(aX, \beta) \leq \ln Z(aX', \beta)$$

This lemma follows from the observation that for any convex function $f$ on the interval $[0, 1]$ we have that $f(x)$ is less than $(1 - x)f(0) + xf(1)$ and so we have the following.

$$\mathrm{E}\left[e^{\beta a X}\right] \leq \mathrm{E}\left[(1 - X) + X e^{\beta a}\right] = (1 - \mathrm{E}[X]) + \mathrm{E}[X] e^{\beta a} = \mathrm{E}\left[e^{\beta a X'}\right]$$

Lemma 11 and equation (2) now imply the following which implies that for the proof of (13) and (14) we can assume without loss of generality that the variables $X_i$ are Bernoulli.

**Lemma 12** *Let $X = \sum_i b_i X_i$ with $X_i \in [0, 1]$ with the variables $X_i$ independent. Let $X' = \sum_i b_i X_i'$ where $X_i' \in \{0, 1\}$ with $\mathrm{E}[X_i'] = \mathrm{E}[X_i]$. For any such $X$, $X'$, and $\epsilon$ we have the following.*
$$S(X, x) \geq S(X', x)$$

.

To prove (13) let $X = \sum_{i=1}^{V} b_i X_i$ where the $X_i$ are independent Bernoulli variables. For $\beta \leq 0$ we have the following.

$$\sigma^2(X_i, \beta) \leq P_\beta(X_i = 1) \leq Q_i$$

So we have $\sigma^2(X, \beta) \leq \sum_i b_i^2 Q_i$. Formula (13) now follows from (9). Formula (14) follows from observations 2 and 3 and the following lemma of Kearns and Saul [13].

**Lemma 13 (Kearns&Saul)** *For a Bernoulli variable $Y$ we have the following where $Q$ is $P(Y = 1)$.*

$$\ln Z(bY, \beta) \quad \leq \quad \mathrm{E}_0[bY]\beta + \frac{(1 - 2Q)b^2}{4\ln\frac{1-Q}{Q}}\beta^2 \qquad (17)$$

$$\leq \quad \mathrm{E}_0[bY]\beta + \frac{b^2}{4\ln\frac{1}{Q}}\beta^2 \qquad (18)$$

# 5 Histogram Rule Error

Now we consider the problem of learning a histogram rule from an IID sample of pairs $\langle x, y \rangle \in \mathcal{X} \times \mathcal{Y}$ drawn from a fixed distribution $D$ on such pairs. The problem is to find a rule $h$ mapping $\mathcal{X}$ to the two-element set $\{0, 1\}$ so as to minimize the expectation of the loss $l(h(x), y)$ where $l$ is a given loss function from $\{0, 1\} \times \mathcal{Y}$ to the interval $[0, 1]$. In the classification setting one typically takes $\mathcal{Y}$ to be $\{0, 1\}$. In the decision-theoretic setting $y$ is the hidden state and can be arbitrarily complex and $l(h(x), y)$ is the cost of taking action $h(x)$ in the presence of hidden state $y$. In the general case (covering both settings) we assume only $h(x) \in \{0, 1\}$ and $\ell(h(x), y) \in [0, 1]$.

We are interested in histogram rules with respect to a fixed clustering. We assume a given cluster function $C$ mapping $\mathcal{X}$ to the integers from 1 to $k$. We consider a sample $S$ of $m$ pairs drawn IID from a fixed distribution on $\mathcal{X} \times \mathcal{Y}$. For any cluster index $j$, we define $S_j$ to be the subset of the sample consisting of pairs $\langle x, y \rangle$ such that $C(x) = j$. We define $\mathrm{c}(j)$ to be $|S_j|$. For any cluster index $j$ and $w \in \{0, 1\}$ we define $l_j(w)$ and $\hat{l}_j(w)$ as follows.

$$\hat{l}_j(w) = \frac{1}{\mathrm{c}(j)} \sum_{\langle x, y \rangle \in S_j} l(w, y), \qquad l_j(w) = \mathrm{E}_{\langle x, y \rangle \sim D \mid C(x) = j} [l(w, y)]$$

If $\mathrm{c}(j) = 0$ then we define $\hat{l}_j(w)$ to be 1. We now define the rule $\hat{h}$ and $h^*$ from class index to labels as follows.

$$\hat{h}(j) = \underset{w \in \{0, 1\}}{\mathrm{argmin}} \ \hat{l}_j(w), \qquad h^*(j) = \underset{w \in \{0, 1\}}{\mathrm{argmin}} \ l_j(w)$$

Ties are broken stochastically with each outcome equally likely so that the rule $h^*$ is a random variable only partially determined by the sample $S$. We are interested in the generalization loss of the empirical rule $\hat{h}$.

$$l(\hat{h}) = \mathrm{E}_{\langle x, y \rangle \sim D} \left[ l(\hat{h}(C(x)), y) \right]$$

**Theorem 14** *For $l(\hat{h})$ defined as above we have the following for positive $\epsilon$.*

$$S\left(l(\hat{h}), \mathrm{E}\left[l(\hat{h})\right] - \epsilon\right) \geq \frac{m\epsilon^2}{7} \tag{19}$$

$$S\left(l(\hat{h}), \mathrm{E}\left[l(\hat{h})\right] + \epsilon\right) \geq \frac{m\epsilon^2}{9} \tag{20}$$

To prove this we need some additional terminology. For each class label $j$ define $P_j$ to be the probability over selecting a pair $\langle x, y \rangle$ that $C(x) = j$. Define $L_j$ to be $l_j(1 - h^*(j)) - l_j(h^*(j))$. In other words, $L_j$ is the additional loss on class $j$ when $\hat{h}$ assigns the wrong label to this class. Define the random variable $X_j$ to be 1 if $\hat{h}(j) \neq h^*(j)$ and 0 otherwise. The variable $X_j$ represents the statement that the empirical rule is "wrong" (non-optimal) on class $j$. We can now express the generalization loss of $\hat{h}$ as follows.

$$l(\hat{h}) = l(h^*) + \sum_i P_i L_i X_i \tag{21}$$

The variable $X_j$ is a monotone stochastic function of the count $\mathrm{c}(j)$ — the probability of error declines monotonically in the count of the class. By lemma 8 we then have that the variables $X_i$ are negatively associated so we can treat them as independent. To prove theorem 14 we start with an analysis of $P(X_j = 1)$.

**Lemma 15**

$$P(X_j = 1) \leq 3e^{-\frac{3}{16} m P_j L_j^2}$$

**Proof:** To prove this lemma we consider a threshold $n \leq mP_j$ and show the following.

$$P(X_j = 1) \quad \leq \quad P(c(j) \leq n) + P(X_j = 1 \mid c(j) \geq n) \tag{22}$$

$$P(c(j) \leq n) \quad \leq \quad e^{-\frac{1}{2}m\left(P_j - \frac{n}{m}\right)^2 / P_j} \tag{23}$$

$$P(X_j = 1 \mid c(j) \geq n) \quad \leq \quad 2e^{-2n\left(\frac{L_j}{2}\right)^2} \tag{24}$$

Formula (23) follows by the Angluin-Valiant bound [1, 7].[1] To prove (24) we note that if $X_j = 1$ then either $\hat{l}_j(h^*(j)) \geq l(h^*(j)) + L/2$ or $\hat{l}_j(1 - h^*(j)) \leq l(1 - h^*(j)) - L/2$. By a combination of Hoeffding's inequality and the union bound we have that the probability that one of these two conditions holds is bounded by the left hand side of (24). Lemma 15 now follows by setting $n$ to $\frac{3}{8}mP_j$ and noting that $L_j \leq 1$. ∎

We now prove (19) using lemma 15 and (10). For $x \leq \mathrm{E}[X]$ we have $\beta(x) \leq 0$ and for $\beta \leq 0$ we have the following.

$$
\begin{aligned}
\sigma^2(P_i^2 L_i^2 X_j, \ \beta) &= \quad P_i^2 L_i^2 P_\beta(X_j = 1)(1 - P_\beta(X_j = 1)) \\
&\leq \quad P_i^2 L_i^2 P_\beta(X_j = 1) \leq P_i^2 L_i^2 P_0(X_j = 1) \leq P_i^2 L_i^2 3e^{-\frac{3}{16}mP_j L_j^2}
\end{aligned}
$$

Since $X_j$ is bounded to the interval $[0, 1]$ we have that $\sigma^2(P_i L_i X_j, \ \beta)$ is also bounded by $P_i^2 L_i^2 / 4$. By (10) we then have the following for $\beta \leq 0$ where $\alpha = (16/3) \ln 12$. In deriving (27) we use the fact that $xe^{-kx}$ is a monotonically decreasing function of $x$ for $x > 1/k$.

$$\ln Z(X, \ \beta) \leq \mathrm{E}_0[X]\beta + \frac{1}{2}\left(\sum_i P_i^2 L_i^2 \min\left(\frac{1}{4}, \ 3e^{-\frac{3}{16}mP_i L_i^2}\right)\right)\beta^2 \tag{25}$$

$$= \mathrm{E}_0[X]\beta + \frac{1}{2}\left(\sum_{mP_i L_i^2 \leq \alpha} \frac{P_i}{m}\left(\frac{mP_i L_i^2}{4}\right) + \sum_{mP_i L_i^2 > \alpha} \frac{P_i}{m}\left(mP_i L_i^2 3e^{-\frac{3}{16}mP_i L_i^2}\right)\right)\beta^2 \tag{26}$$

$$\leq \mathrm{E}_0[X]\beta + \frac{1}{2}\left(\sum_{mP_i L_i^2 \leq \alpha} \frac{P_i}{m}\left(\frac{\alpha}{4}\right) + \sum_{mP_i L_i^2 > \alpha} \frac{P_i}{m}\left(3\alpha e^{-\frac{3}{16}\alpha}\right)\right)\beta^2 \tag{27}$$

$$= \mathrm{E}_0[X]\beta + \frac{1}{2}\left(\sum_i \frac{P_i}{m}\left(\frac{\alpha}{4}\right)\right)\beta^2 \tag{28}$$

$$= \mathrm{E}_0[X]\beta + \frac{1}{2}\left(\frac{\alpha}{4m}\right)\beta^2 \tag{29}$$

Formula (19) now follows from (29) and a downward variant of observation 2. The proof of (20) is similar but uses (18). For $\beta \geq 0$ we have the following where $\alpha$ is $16(2 + \ln 3)/3$.

$$\ln Z(X, \ \beta) \quad \leq \quad \mathrm{E}_0[X]\beta + \frac{1}{2}\left(\sum_{mP_i L_i^2 \leq \alpha} \frac{P_i}{m}\left(\frac{mP_i L_i^2}{4}\right) + \sum_{mP_i L_i^2 > \alpha} \frac{P_i}{m}\left(\frac{mP_i L_i^2}{2\left(\frac{3}{16}mP_i L_i^2 - \ln 3\right)}\right)\right)\beta^2$$

$$\leq \quad \mathrm{E}_0[X]\beta + \frac{1}{2}\left(\sum_{mP_i L_i^2 \leq \alpha} \frac{P_i}{m}\left(\frac{\alpha}{4}\right) + \sum_{mP_i L_i^2 > \alpha} \frac{P_i}{m}\left(\frac{1}{2\left(\frac{3}{16} - \frac{\ln 3}{\alpha}\right)}\right)\right)\beta^2$$

$$= \mathrm{E}_0\left[X\right]\beta + \frac{1}{2}\left(\sum_i \frac{P_i}{m}\left(\frac{\alpha}{4}\right)\right)\beta^2 \qquad (30)$$

$$= \mathrm{E}_0\left[X\right]\beta + \frac{1}{2}\left(\frac{\alpha}{4m}\right)\beta^2 \qquad (31)$$

Formula (20) now follows from (31) and observation 2.

## Footnotes

[1] The downward deviation Angluin-Valiant bound used here follows from (9) and the observation that for a Bernoulli variable $Y$ and $\beta \leq 0$ we have $\sigma^2(Y, \ \beta) \leq P(Y = 1)$.

## References

[1] D. Anguluin and L. Valiant. Fast probabalistic algorithms for hamiltonian circuits. *Journal of Computing Systems Science*, 18:155–193, 1979.

[2] G. Bennnett. Probability inequalities for the sum of independent ranndom variables. *Journal of the American Statistical Association*, 57:33–45, 1962.

[3] S. Bernstein. *The Theory of Probabilities*. Gastehizdat Publishing House, Moscow, 1946.

[4] Stanley Chen and Joshua Goodman. An empirical study of smoothing techniques for language modeling, August 1998. Technical report TR-10-98, Harvard University.

[5] H. Chernoff. A measure of the asymptotic effi ciency of tests of a hypothesis based on the sum of observations. *Annals of Mathmematical Statistics*, 23:493–507, 1952.

[6] Kenneth W. Church and William A. Gale. A comparison of the enhanced Good-Turing and deleted estimation methods for estimating probabilities of English bigrams. *Computer Speech and Language*, 5:19–54, 1991.

[7] Luc Devroye, László Györfi , and Gábor Lugosi. *A Probabilistic Theory of Pattern Recognition*. Springer, 1996.

[8] Devdatt P. Dubhashi and Desh Ranjan. Balls and bins: A study in negative dependence. *Random Structures and Algorithms*, 13(2):99–124, 1998.

[9] I. J. Good. The population frequencies of species and the estimation of population parameters. *Biometrika*, 40(16):237–264, December 1953.

[10] T. Hagerup and C. Rüb. A guided tour of chernoff bounds. *Information Processing Letters*, 33:305–309, 1989.

[11] W. Hoeffding. Probability inequalities for sums of bounded random variables. *Journal of the American Statistical Association*, 58:13–30, 1963.

[12] Slava M. Katz. Estimation of probabilities from sparse data for the language model component of a speech recognizer. *IEEE Transactions on Acoustics, Speech and Signal Processing*, ASSP-35(3):400–401, March 1987.

[13] Michael Kearns and Lawrence Saul. Large deviation methods for approximate probabilistic inference, with rates of convergence. In *UAI-98*, pages 311–319. Morgan Kaufmann, 1998.

[14] Samuel Kutin. *Algorithmic Stability and Ensemble-Based Learning*. PhD thesis, University of Chicago, 2002.

[15] David McAllester and Robert Schapire. On the convergence rate of good-turing estimators. In *COLT00*, 2000.

[16] Luis E. Ortiz and Leslie Pack Kaelbling. Sampling methods for action selection in influence diagrams. In *Proceedings of the Seventeenth National Conference on Artificial Intelligence*, pages 378–385, 2000.

[17] Anand Srivastav and Peter Stangier. Integer multicommodity flows with reduced demands. In *European Symposium on Algorithms*, pages 360–371, 1993.
